# Supervised Learning of Probability Distributions
# by Neural Networks

Eric B. Baum

Jet Propulsion Laboratory, Pasadena CA 91109

Frank Wilczek[†]

Department of Physics,Harvard University,Cambridge MA 02138

**Abstract:**

We propose that the back propagation algorithm for supervised learning can be generalized, put on a satisfactory conceptual footing, and very likely made more efficient by defining the values of the output and input neurons as probabilities and varying the synaptic weights in the gradient direction of the log likelihood, rather than the 'error'.

In the past thirty years many researchers have studied the question of supervised learning in 'neural'-like networks. Recently a learning algorithm called 'back propagation'[1−4] or the 'generalized delta-rule' has been applied to numerous problems including the mapping of text to phonemes[5], the diagnosis of illnesses[6] and the classification of sonar targets[7]. In these applications, it would often be natural to consider imperfect, or probabilistic information. We believe that by considering supervised learning from this slightly larger perspective, one can not only place back propaga-

† Permanent address: Institute for Theoretical Physics, University of California, Santa Barbara CA 93106

tion on a more rigorous and general basis, relating it to other well studied pattern recognition algorithms, but very likely improve its performance as well.

The problem of supervised learning is to model some mapping between input vectors and output vectors presented to us by some real world phenomena. To be specific, consider the question of medical diagnosis. The input vector corresponds to the symptoms of the patient; the i-th component is defined to be 1 if symptom i is present and 0 if symptom i is absent. The output vector corresponds to the illnesses, so that its j-th component is 1 if the j-th illness is present and 0 otherwise. Given a data base consisting of a number of diagnosed cases, the goal is to construct (learn) a mapping which accounts for these examples and can be applied to diagnose new patients in a reliable way. One could hope, for instance, that such a learning algorithm might yield an expert system to simulate the performance of doctors. Little expert advice would be required for its design, which is advantageous both because experts' time is valuable and because experts often have extraodinary difficulty in describing how they make decisions.

A feedforward neural network implements such a mapping between input vectors and output vectors. Such a network has a set of input nodes, one or several layers of intermediate nodes, and a layer of output nodes. The nodes are connected in a forward directed manner, so that the output of a node may be connected to the inputs of nodes in subsequent layers, but closed loops do not occur. See figure 1. The output of each node is assumed to be a bounded semilinear function of its inputs. That is, if $v_j$ denotes the output of the j-th node and $w_{ij}$ denotes the weight associated with the connection of the output of the j-th node to the input of

the i-th, then the i-th neuron takes value $v_i = g(\sum_j w_{ij} v_j)$, where g is a bounded, differentiable function called the activation function. $g(x) = 1/(1 + e^{-x})$, called the logistic function, is frequently used. Given a fixed set of weights $\{w_{ij}\}$, we set the input node values to equal some input vector, compute the value of the nodes layer by layer until we compute the output nodes, and so generate an output vector.

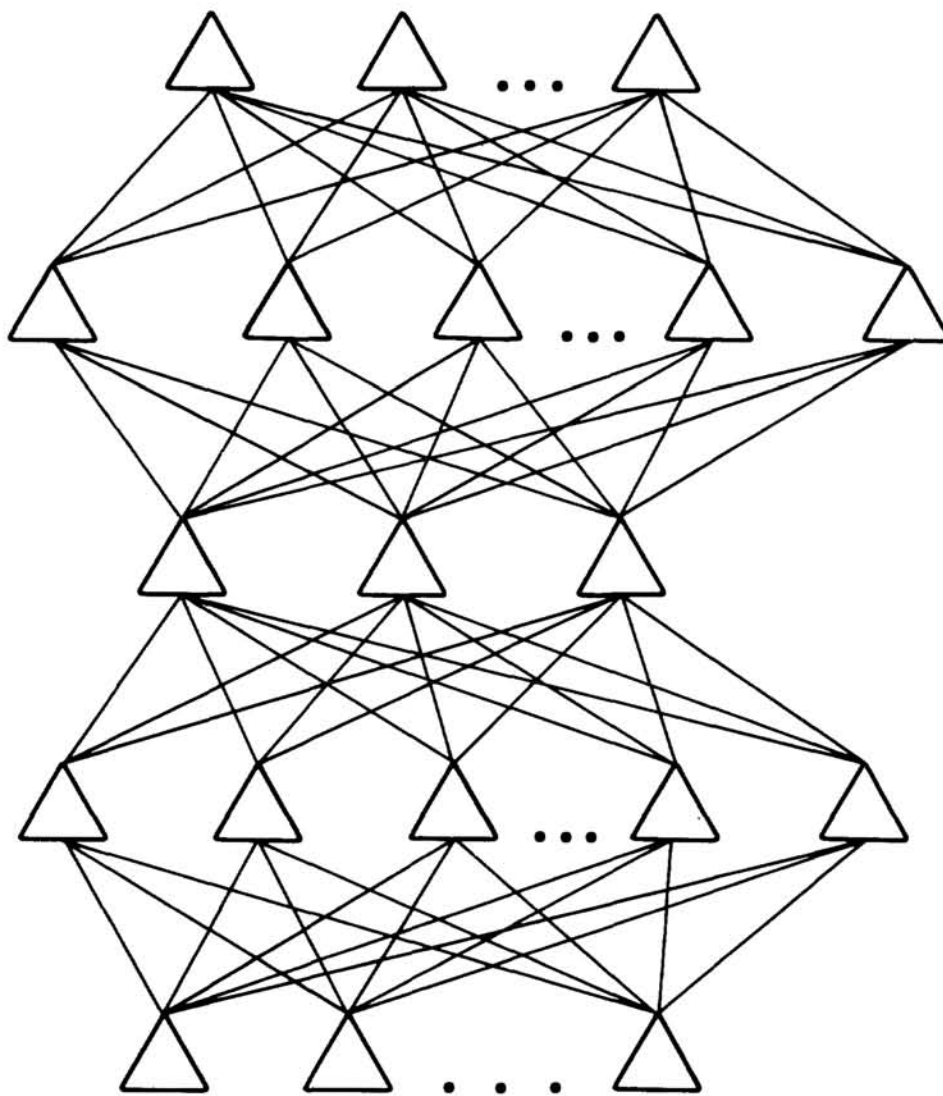

**Figure 1: A 5 layer network. Note bottleneck at layer 3.**

Such networks have been studied because of analogies to neurobiology, because it may be easy to fabricate them in hardware, and because learning algorithms such as the Perceptron learning algorithm[8], Widrow- Hoff[9], and backpropagation have been able to choose weights $w_{ij}$ that solve interesting problems.

Given a set of input vectors $s_i^\mu$, together with associated target values $t_j^\mu$, back propagation attempts to adjust the weights so as to minimize the error E in achieving these target values, defined as

$$E = \sum_\mu E_\mu = \sum_{\mu,j} (t_j^\mu - o_j^\mu)^2 \qquad (1)$$

where $o_j^\mu$ is the output of the j-th node when $s^\mu$ is presented as input. Back propagation starts with randomly chosen $w_{ij}$ and then varies in the gradient direction of E until a local minimum is obtained. Although only a locally optimal set of weights is obtained, in a number of experiments the neural net so generated has performed surprisingly well not only on the training set but on subsequent data.[4-6] This performance is probably the main reason for widespread interest in backpropagation.

It seems to us natural, in the context of the medical diagnosis problem, the other real world problems to which backpropagation has been applied, and indeed in any mapping problem where one desires to generalize from a limited and noisy set of examples, to interpret the output vector in probabilistic terms. Such an interpretation is standard in the literature on pattern classification.[10] Indeed, the examples might even be probabilistic themselves. That is to say it might not be certain whether symptom $i$ was present in case $\mu$ or not.

Let $s_i^\mu$ represent the probability symptom i is present in case $\mu$, and let $t_j^\mu$ represent the probability disease $j$ occurred in case

$\mu$. Consider for the moment the case where the $t_j^\mu$ are 1 or 0, so that the cases are in fact fully diagnosed. Let $f_j(\hat{s}, \hat{\theta})$ be our prediction of the probability of disease $j$ given input vector $\hat{s}$, where $\hat{\theta}$ is some set of parameters determined by our learning algorithm. In the neural network case, the $\hat{\theta}$ are the connection weights and $f_j(\hat{s}^\mu, \{\hat{w}_{ij}\}) = o_j^\mu$.

Now lacking a priori knowledge of good $\hat{\theta}$, the best one can do is to choose the parameters $\hat{\theta}$ to maximize the likelihood that the given set of examples should have occurred.[10] The formula for this likelihood, p, is immediate:

$$p = \prod_\mu \left[ \prod_{\{j|t_j^\mu=1\}} f_j(\hat{s}^\mu, \hat{\theta}) \prod_{\{j|t_j^\mu=0\}} (1 - f_j(\hat{s}^\mu, \hat{\theta})) \right] \qquad (2)$$

or

$$log(p) = \sum_\mu \left[ \sum_{\{j|t_j^\mu=1\}} log(f_j(\hat{s}^\mu, \hat{\theta})) + \sum_{\{j|t_j^\mu=0\}} log(1 - f_j(\hat{s}^\mu, \hat{\theta})) \right]$$
$$(3)$$

The extension of equation (2), and thus equation (3) to the case where the $\hat{t}$ are probabilities, taking values in $[0, 1]$, is straight-

forward[*1] and yields

$$log(p) = \sum_{\mu,j} \left[ t_j^\mu log(f_j(\hat{s}^\mu, \hat{\theta})) + (1 - t_j^\mu)log(1 - f_j((\hat{s}^\mu, \hat{\theta})) \right] \quad (4)$$

Expressions of this sort often arise in physics and information theory and are generally interpreted as an entropy.[11]

We may now vary the $\{\hat{\theta}\}$ in the gradient direction of the entropy. The back propagation algorithm generalizes immediately from minimizing 'Error' or 'Energy' to maximizing entropy or log likelihood, or indeed any other function of the outputs and the inputs[12]. Of course it remains true that the gradient can be computed by back propagation with essentially the same number of computations as are required to compute the output of the network.

A backpropagation algorithm based on log-likelihood is not only more intuitively appealing than one based on an ad-hoc definition of error, but will make quite different and more accurate predictions as well. Consider e.g. training the net on an example which it already understands fairly well. Say $t_j^o = 0$, and $f_j(s^o) = \epsilon$. Now, from eqn(1) $\partial E/\partial f_j = 2\epsilon$, so using 'Error' as a

---

[*1] We may see this by constructing an equivalent larger set of examples with the $\hat{t}$ taking only values 0 or 1 with the appropriate frequency. Thus assume the $t_j^\mu$ are rational numbers with denominator $d_j^\mu$ and numerator $n_j^\mu$ and let $p = \prod_{\mu,j} d_j^\mu$. What we mean by the set of examples $\{t^\mu : \mu = 1, ..., M\}$ can be represented by considering a set of $N = Mp$ examples $\{\tilde{t}_j^\nu\}$ where for each $\mu$, $\tilde{t}_j^\nu = 0$ for $p(\mu - 1) < \nu \leq p\mu$ and $1 \leq \nu mod(d_j^\mu) \leq (d_j^\mu - n_j^\mu)$, and $\tilde{t}_j^\nu = 1$ otherwise. Now applying equation (3) gives equation (4), up to an overall normalization.

criterion the net learns very little from this example, whereas, using eqn(3), $\partial log(p)/\partial f_j = 1/(1 - \epsilon)$, so the net continues to learn and can in fact converge to predict probabilities near 1. Indeed because backpropagation using the standard 'Error' measure can not converge to generate outputs of 1 or 0, it has been customary in the literature[4] to round the target values so that a target of 1 would be presented in the learning algorithm as some ad hoc number such as .8, whereas a target of 0 would be presented as .2.

In the context of our general discussion it is natural to ask whether using a feedforward network and varying the weights is in fact the most effective alternative. Anderson and Abrahams[13] have discussed this issue from a Bayesian viewpoint. From this point of view, fitting output to input using normal distributions and varying the means and covariance matrix may seem to be more logical.

Feedforward networks do however have several advantages for complex problems. Experience with neural networks has shown the importance of including hidden units wherein the network can form an internal representation of the world. If one simply uses normal distributions, any hidden variables included will simply integrate out in calculating an output. It will thus be necessary to include at least third order correlations to implement useful hidden variables. Unfortunately, the number of possible third order correlations is very large, so that there may be practical obstacles to such an approach. Indeed it is well known folklore in curve fitting and pattern classification that the number of parameters must be small compared to the size of the data set if any generalization to future cases is expected.[10]

In feedforward nets the question takes a different form. There can be bottlenecks to information flow. Specifically, if the net is

constructed with an intermediate layer which is not bypassed by any connections (i.e. there are no connections from layers preceding to layers subsequent), and if furthermore the activation functions are chosen so that the values of each of the intermediate nodes tend towards either 1 or 0*2, then this layer serves as a bottleneck to information flow. No matter how many input nodes, output nodes, or free parameters there are in the net, the output will be constrained to take on no more than $2^I$ different patterns, where I is the number of nodes in the bottleneck layer. Thus if I is small, some sort of 'generalization' must occur even if the number of weights is large. One plausible reason for the success of back propagation in adequately solving tasks, in spite of the fact that it finds only local minima, is its ability to vary a large number of parameters. This freedom may allow back propagation to escape from many putative traps and to find an acceptable solution.

A good expert system, say for medical diagnosis, should not only give a diagnosis based on the available information, but should be able to suggest, in questionable cases, which lab tests might be performed to clarify matters. Actually back propagation inherently has such a capability. Back propagation involves calculation of $\partial log(p)/\partial w_{ij}$. This information allows one to compute immediately $\partial log(p)/\partial s_j$. Those input nodes for which this partial derivative is large correspond to important experiments.

In conclusion, we propose that back propagation can be generalized, put on a satisfactory conceptual footing, and very likely made more efficient, by defining the values of the output and in-

put neurons as probabilities, and replacing the 'Error' by the log-likelihood.

*Acknowledgement:* E. B. Baum was supported in part by DARPA through arrangement with NASA and by NSF grant DMB-840649, 802. F. Wilczek was supported in part by NSF grant PHY82-17853

## Footnotes

*2 Alternatively when necessary this can be enforced by adding an energy term to the log-likelihood to constrain the parameter variation so that the neuronal values are near either 1 or 0.

## References

(1)Werbos,P,"Beyond Regression: New Tools for Prediction and Analysis in the Behavioral Sciences", Harvard University Dissertation (1974)

(2)Parker D. B.,"Learning Logic",MIT Tech Report TR-47, Center for Computationl Research in Economics and Management Science, MIT, 1985

(3)Le Cun, Y., Proceedings of Cognitiva 85,p599-604, Paris (1985)

(4)Rumelhart, D. E., Hinton, G. E., Williams, G. E., "Learning Internal Representations by Error Propagation", in "Parallel Distributed Processing", vol 1, eds. Rumelhart, D. E., McClelland, J. L., MIT Press, Cambridge MA,( 1986)

(5)Sejnowski, T. J., Rosenberg, C. R., Complex Systems, v 1, pp 145-168 (1987)

(6)LeCun, Y., Address at 1987 Snowbird Conference on Neural Networks

(7)Gorman, P., Sejnowski, T. J.,"Learned Classification of Sonar Targets Using a Massively Parallel Network", in "Workshop on Neural Network Devices and Applications", JPLD-4406, (1987) pp224-237

(8)Rosenblatt, F.,"Principles of Neurodynamics: Perceptrons and

the theory of brain mechanisms", Spartan Books, Washington DC (1962)

(9)Widrow, B., Hoff, M. E., 1960 IRE WESCON Conv. Record, Part 4, 96-104 (1960)

(10)Duda, R. O., Hart, P. E., "Pattern Classification and Scene Analysis", John Wiley and Sons, N.Y., (1973)

(11)Guiasu, S., "Information Theory with Applications", McGraw Hill, NY, (1977)

(12)Baum,E.B.,"Generalizing Back Propagation to Computation", in "Neural Networks for Computing", AIP Conf. Proc. 151, Snowbird UT (1986)pp47-53

(13)Anderson, C.H., Abrahams, E.,"The Bayes Connection", Proceedings of the IEEE International Conference on Neural Networks, San Diego,(1987)
